# Self-Organizing and Adaptive Algorithms for Generalized Eigen-Decomposition

**Chanchal Chatterjee**
Newport Corporation
1791 Deere Avenue, Irvine, CA 92606

**Vwani P. Roychowdhury**
Electrical Engineering Department
UCLA, Los Angeles, CA 90095

## ABSTRACT

The paper is developed in two parts where we discuss a new approach to self-organization in a single-layer linear feed-forward network. First, two novel algorithms for self-organization are derived from a two-layer linear hetero-associative network performing a one-of-$m$ classification, and trained with the constrained least-mean-squared classification error criterion. Second, two adaptive algorithms are derived from these self-organizing procedures to compute the principal generalized eigenvectors of two correlation matrices from two sequences of random vectors. These novel adaptive algorithms can be implemented in a single-layer linear feed-forward network. We give a rigorous convergence analysis of the adaptive algorithms by using *stochastic approximation theory*. As an example, we consider a problem of online signal detection in digital mobile communications.

## 1. INTRODUCTION

We study the problems of hetero-associative training, linear discriminant analysis, generalized eigen-decomposition and their theoretical connections. The paper is divided into two parts. In the first part, we study the relations between hetero-associative training with a linear feed-forward network, and feature extraction by the linear discriminant analysis (LDA) criterion. Here we derive two novel algorithms that unify the two problems. In the second part, we generalize the self-organizing algorithm for LDA to obtain adaptive algorithms for generalized eigen-decomposition, for which we provide a rigorous proof of convergence by using *stochastic approximation theory*.

### 1.1 HETERO-ASSOCIATION AND LINEAR DISCRIMINANT ANALYSIS

In this discussion, we consider a special case of hetero-association that deals with the classification problems. Here the inputs belong to a finite $m$-set of pattern classes, and the

outputs indicate the classes to which the inputs belong. Usually, the $i^{th}$ standard basis vector $e_i$ is chosen to indicate that a particular input vector $x$ belongs to class $i$.

The LDA problem, on the other hand, aims at projecting a multi-class data in a lower dimensional subspace such that it is grouped into well-separated clusters for the $m$ classes. The method is based upon a set of scatter matrices commonly known as the mixture scatter $S_m$ and between class scatter $S_b$ (Fukunaga, 1990). These matrices are used to formulate criteria such as $\text{tr}(S_m^{-1}S_b)$ and $\det(S_b)/\det(S_m)$ which yield a linear transform $\Phi$ that satisfy the generalized eigenvector problem $S_b\Phi=S_m\Phi\Lambda$, where $\Lambda$ is the generalized eigenvalue matrix. If $S_m$ is positive definite, we obtain a $\Phi$ such that $\Phi^T S_m \Phi =I$ and $\Phi^T S_b \Phi=\Lambda$. Furthermore, the significance of each eigenvector (for class separability) is determined by the corresponding generalized eigenvalue.

A relation between hetero-association and LDA was demonstrated by Gallinari *et al.* (1991). Their work made explicit that for a linear multi-layer perceptron performing a one-from-$m$ classification that minimized the total mean square error (MSE) at the network output, also maximized a criterion $\det(S_b)/\det(S_m)$ for LDA at the final hidden layer. This study was generalized by Webb and Lowe (1990) by using a nonlinear transform from the input data to the final hidden units, and a linear transform in the final layer. This has been further generalized by Chatterjee and Roychowdhury (1996) by including the Bayes cost for misclassification into the criteria $\text{tr}(S_m^{-1}S_b)$.

Although the above studies offer useful insights into the relations between hetero-association and LDA, they do not suggest an *algorithm* to extract the optimal LDA transform $\Phi$. Since the criteria for class separability are insensitive to multiplication by nonsingular matrices, the above studies suggest that any training procedure that minimizes the MSE at the network output will yield a nonsingular transformation of $\Phi$; i.e., we obtain $Q\Phi$ where $Q$ is a nonsingular matrix. Since $Q\Phi$ does not satisfy the generalized eigenvector problem $S_b\Phi=S_m\Phi\Lambda$ for any arbitrary nonsingular matrix $Q$, we need to determine an algorithm that will yield $Q=I$.

In order to obtain the optimum linear transform $\Phi$, we constrain the training of a two-layer linear feed-forward network, such that at convergence, the weights for the first layer simultaneously diagonalizes $S_m$ and $S_b$. Thus, the hetero-associative network is trained by minimizing a *constrained MSE* at the network output. This training procedure yields two novel algorithms for LDA.

## 1.2 LDA AND GENERALIZED EIGEN-DECOMPOSITION

Since the LDA problem is a generalized eigen-decomposition problem for the symmetric-definite case, the self-organizing algorithms derived from the hetero-associative networks lead us to construct *adaptive algorithms* for generalized eigen-decomposition. Such adaptive algorithms are required in several applications of image and signal processing. As an example, we consider the problem of online interference cancellation in digital mobile communications.

Similar to the LDA problem $S_b\Phi=S_m\Phi\Lambda$, the generalized eigen-decomposition problem $A\Phi=B\Phi\Lambda$ involves the matrix pencil $(A,B)$, where $A$ and $B$ are assumed to be real, symmetric and positive definite. Although a solution to the problem can be obtained by a conventional method, there are several applications in image and signal processing where an online solution of generalized eigen-decomposition is desired. In these real-time situations, the matrices $A$ and $B$ are themselves unknown. Instead, there are available two

sequences of random vectors $\{\mathbf{x}_k\}$ and $\{\mathbf{y}_k\}$ with $\lim_{k\to\infty}E[\mathbf{x}_k\mathbf{x}_k^T]=A$ and $\lim_{k\to\infty}$ $E[\mathbf{y}_k\mathbf{y}_k^T]=B$, where $\mathbf{x}_k$ and $\mathbf{y}_k$ represent the online observations of the application. For every sample $(\mathbf{x}_k,\mathbf{y}_k)$, we need to obtain the current estimates $\Phi_k$ and $\Lambda_k$ of $\Phi$ and $\Lambda$ respectively, such that $\Phi_k$ and $\Lambda_k$ converge strongly to their true values.

The conventional approach for evaluating $\Phi$ and $\Lambda$ requires the computation of $(A,B)$ after collecting all of the samples, and then the application of a numerical procedure; i.e., the approach works in a *batch* fashion. There are two problems with this approach. Firstly, the dimension of the samples may be large so that even if all of the samples are available, performing the generalized eigen-decomposition may take prohibitively large amount of computational time. Secondly, the conventional schemes can not adapt to slow or small changes in the data. So the approach is not suitable for real-time applications where the samples come in an *online* fashion.

Although the adaptive generalized eigen-decomposition algorithms are natural generalizations of the self-organizing algorithms for LDA, their derivations do not constitute a proof of convergence. We, therefore, give a rigorous proof of convergence by *stochastic approximation theory*, that shows that the estimates obtained from our adaptive algorithms converge with probability one to the generalized eigenvectors.

In summary, the study offers the following contributions: (1) we present two novel algorithms that unify the problems of hetero-associative training and LDA feature extraction; and (2) we discuss two single-stage adaptive algorithms for generalized eigen-decomposition from two sequences of random vectors.

In our experiments, we consider an example of online interference cancellation in digital mobile communications. In this problem, the signal from a desired user at a far distance from the receiver is corrupted by another user very near to the base. The optimum linear transform $\mathbf{w}$ for weighting the signal is the first principal generalized eigenvector of the signal correlation matrix with respect to the interference correlation matrix. Experiments with our algorithm suggest a rapid convergence within four bits of transmitted signal, and provides a significant advantage over many current methods.

## 2. HETERO-ASSOCIATIVE TRAINING AND LDA

We consider a two-layer linear network performing a one-from-$m$ classification. Let $\mathbf{x}\in\Re^n$ be an input to the network to be classified into one out of $m$ classes $\omega_1,...,\omega_m$. If $\mathbf{x}\in\omega_i$ then the desired output $\mathbf{d}=\mathbf{e}_i$ ($i^{th}$ std. basis vector). Without loss of generality, we assume the inputs to be a zero-mean stationary process with a nonsingular covariance matrix.

### 2.1 EXTRACTING THE PRINCIPAL LDA COMPONENTS

In the two-layer linear hetero-associative network, let there be $p$ neurons in the hidden layer, and $m$ output units. The aim is to develop an algorithm so that individual weight vectors for the first layer converge to the first $p\leq m$ generalized eigenvectors corresponding to the $p$ significant generalized eigenvalues arranged in decreasing order. Let $\mathbf{w}_i\in\Re^n$ ($i=1,...,n$) be the weight vectors for the input layer, and $\mathbf{v}_i\in\Re^m$ ($i=1,...,m$) be the weight vectors for the output layer.

The neurons are trained sequentially; i.e., the training of the $j^{th}$ neuron is started only after the weight vector of the $(j-1)^{th}$ neuron has converged. Assume that all the $j-1$ previous neurons have already been trained and their weights have converged to the

optimal weight vectors $\mathbf{w}_i$ for $i \in [1, j-1]$. To extract the $j^{th}$ generalized eigenvector in the output of the $j^{th}$ neuron, the updating model for this neuron should be constructed by subtracting the results from all previously computed $j-1$ generalized eigenvectors from the desired output $\mathbf{d}_j$ as below

$$\tilde{\mathbf{d}}_j = \mathbf{d}_j - \sum_{i=1}^{j-1} \mathbf{v}_i \mathbf{w}_i^T \mathbf{x}. \tag{1}$$

This process is equivalent to the *deflation* of the desired output.

The scatter matrices $S_m$ and $S_b$ can be obtained from $\mathbf{x}$ and $\mathbf{d}$ as $S_m = E[\mathbf{x}\mathbf{x}^T]$ and $S_b = MM^T$, where $M = E[\mathbf{x}\mathbf{d}^T]$. We need to extract the $j^{th}$ LDA transform $\mathbf{w}_j$ that satisfies the generalized eigenvector equation $S_b\mathbf{w}_j = \lambda_j S_m \mathbf{w}_j$ such that $\lambda_j$ is the $j^{th}$ largest generalized eigenvalue. The constrained MSE criterion at the network output is

$$J\left(\mathbf{w}_j, \mathbf{v}_j\right) = E\left[\left\| \mathbf{d}_j - \sum_{i=1}^{j-1} \mathbf{v}_i \mathbf{w}_i^T \mathbf{x} - \mathbf{v}_j \mathbf{w}_j^T \mathbf{x} \right\|^2\right] + \mu(\mathbf{w}_j^T S_m \mathbf{w}_j - 1). \tag{2}$$

Using (2), we obtain the update equation for $\mathbf{w}_j$ as

$$\mathbf{w}_{k+1}^{(j)} = \mathbf{w}_k^{(j)} + \eta\left( M\mathbf{v}_k^{(j)} - S_m \mathbf{w}_k^{(j)}\left(\mathbf{w}_k^{(j)T} M\mathbf{v}_k^{(j)}\right) - S_m \sum_{i=1}^{j-1} \mathbf{w}_k^{(j)} \mathbf{v}_k^{(i)T} \mathbf{v}_k^{(j)}\right). \tag{3}$$

Differentiating (2) with respect to $\mathbf{v}_j$, and equating it to zero, we obtain the optimum value of $\mathbf{v}_j$ as $M^T\mathbf{w}_j$. Substituting this $\mathbf{v}_j$ in (3) we obtain

$$\mathbf{w}_{k+1}^{(j)} = \mathbf{w}_k^{(j)} + \eta\left( S_b \mathbf{w}_k^{(j)} - S_m \mathbf{w}_k^{(j)}\left(\mathbf{w}_k^{(j)T} S_b \mathbf{w}_k^{(j)}\right) - S_m \sum_{i=1}^{j-1} \mathbf{w}_k^{(i)} \mathbf{w}_k^{(i)T} S_b \mathbf{w}_k^{(j)}\right). \tag{4}$$

Let $W_k$ be the matrix whose $i^{th}$ column is $\mathbf{w}_k^{(i)}$. Then (4) can be written in matrix form as

$$W_{k+1} = W_k + \eta\left( S_b W_k - S_m W_k \mathrm{UT}\left[W_k^T S_b W_k\right]\right), \tag{5}$$

where $\mathrm{UT}[\cdot]$ sets all elements below the diagonal of its matrix argument to zero, thereby making it upper triangular.

## 2.2 ANOTHER SELF-ORGANIZING ALGORITHM FOR LDA

In the previous analysis for a two-layer linear hetero-associative network, we observed that the optimum value for $V = W^T M$, where the $i^{th}$ column of $W$ and row of $V$ are formed by $\mathbf{w}_i$ and $\mathbf{v}_i$ respectively. It is, therefore, worthwhile to explore the gradient descent procedure on the error function below instead of (2)

$$J(W) = E\left[\left\| \mathbf{d} - M^T W W^T \mathbf{x} \right\|^2\right]. \tag{6}$$

By differentiating this error function with respect to $W$, and including the deflation process, we obtain the following update procedure for $W$ instead of (5)

$$W_{k+1} = W_k + \eta\left(2S_b W_k - S_m W_k \mathrm{UT}\left[W_k^T S_b W_k\right] - S_b W_k \mathrm{UT}\left[W_k^T S_m W_k\right]\right). \tag{7}$$

## 3. LDA AND GENERALIZED EIGEN-DECOMPOSITION

Since LDA consists of solving the generalized eigenvector problem $S_b\Phi = S_m\Phi\Lambda$, we can naturally generalize algorithms (5) and (7) to obtain adaptive algorithms for the generalized eigen-decomposition problem $A\Phi = B\Phi\Lambda$, where $A$ and $B$ are assumed to be symmetric and positive definite. Here, we do not have the matrices $A$ and $B$. Instead,

there are available two sequences of random vectors $\{\mathbf{x}_k\}$ and $\{\mathbf{y}_k\}$ with $\lim_{k\to\infty}E[\mathbf{x}_k\mathbf{x}_k^T]$ $=A$ and $\lim_{k\to\infty}E[\mathbf{y}_k\mathbf{y}_k^T]=B$, where $\mathbf{x}_k$ and $\mathbf{y}_k$ represent the online observations.

From (5), we obtain the following adaptive algorithm for generalized eigen-decomposition

$$W_{k+1} = W_k + \eta_k\Big(A_kW_k - B_kW_k\mathrm{UT}\big[W_k^TA_kW_k\big]\Big). \tag{8}$$

Here $\{\eta_k\}$ is a sequence of scalar gains, whose properties are described in Section 4. The sequences $\{A_k\}$ and $\{B_k\}$ are instantaneous values of the matrices $A$ and $B$ respectively. Although the $A_k$ and $B_k$ values can be obtained from $\mathbf{x}_k$ and $\mathbf{y}_k$ as $\mathbf{x}_k\mathbf{x}_k^T$ and $\mathbf{y}_k\mathbf{y}_k^T$ respectively, our algorithm requires that at least one of the $\{A_k\}$ or $\{B_k\}$ sequences have a dominated convergence property. Thus, the $\{A_k\}$ and $\{B_k\}$ sequences may be obtained from $\mathbf{x}_k\mathbf{x}_k^T$ and $\mathbf{y}_k\mathbf{y}_k^T$ from the following algorithms

$$A_k = A_{k-1} + \gamma_k\big(\mathbf{x}_k\mathbf{x}_k^T - A_{k-1}\big) \text{ and } B_k = B_{k-1} + \gamma_k\big(\mathbf{y}_k\mathbf{y}_k^T - B_{k-1}\big), \tag{9}$$

where $A_0$ and $B_0$ are symmetric, and $\{\gamma_k\}$ is a scalar gain sequence.

As done before, we can generalize (7) to obtain the following adaptive algorithm for generalized eigen-decomposition from a sequence of samples $\{A_k\}$ and $\{B_k\}$

$$W_{k+1} = W_k + \eta_k\Big(2A_kW_k - B_kW_k\mathrm{UT}\big[W_k^TA_kW_k\big] - A_kW_k\mathrm{UT}\big[W_k^TB_kW_k\big]\Big). \tag{10}$$

Although algorithms (8) and (10) were derived from the network MSE by the gradient descent approach, this derivation does not guarantee their convergence. In order to prove their convergence, we use *stochastic approximation theory*. We give the convergence results only for algorithm (10).

## 4. STOCHASTIC APPROX. CONVG. PROOF FOR ALG. (10)

In order to prove the convergence of (10), we use stochastic approximation theory due to Ljung (1977). In stochastic approximation theory, we study the asymptotic properties of (10) in terms of the ordinary differential equation (ODE)

$$\frac{d}{dt}W(t) = \lim_{k\to\infty} E\Big[2A_kW - B_kW\mathrm{UT}\big[W^TA_kW\big] - A_kW\mathrm{UT}\big[W^TB_kW\big]\Big],$$

where $W(t)$ is the continuous time counterpart of $W_k$ with $t$ denoting continuous time. The method of proof requires the following steps: (1) establishing a set of conditions to be imposed on $A$, $B$, $A_k$, $B_k$, and $\eta_k$, (2) finding the stable stationary points of the ODE; and (3) demonstrating that $W_k$ visits a compact subset of the domain of attraction of a stable stationary point infinitely often.

We use Theorem 1 of Ljung (1977) for the convergence proof. The following is a general set of assumptions for the convergence proof of (10):

**Assumption (A1).** Each $\mathbf{x}_k$ and $\mathbf{y}_k$ is bounded with probability one, and $\lim_{k\to\infty}E[\mathbf{x}_k\mathbf{x}_k^T]$ $=A$ and $\lim_{k\to\infty}E[\mathbf{y}_k\mathbf{y}_k^T] = B$, where $A$ and $B$ are positive definite.

**Assumption (A2).** $\{\eta_k\in\Re^+\}$ satisfies $\eta_k\downarrow0$, $\sum_{k=0}^{\infty}\eta_k=\infty$, $\sum_{k=0}^{\infty}\eta_k^r<\infty$ for some $r>1$ and $\lim_{k\to\infty}\sup(\eta_k^{-1} - \eta_{k-1}^{-1})<\infty$.

**Assumption (A3).** The $p$ largest generalized eigenvalues of $A$ with respect to $B$ are each of unit multiplicity.

**Lemma 1.** *Let A1 and A2 hold. Let $W^*$ be a locally asymptotically stable (in the sense of Liapunov) solution to the ordinary differential equation (ODE):*

$$\frac{d}{dt}W(t) = 2AW(t) - BW(t)\mathrm{UT}\left[W(t)^T AW(t)\right] - AW(t)\mathrm{UT}\left[W(t)^T BW(t)\right], \qquad (11)$$

*with domain of attraction $D(W^*)$. Then if there is a compact subset $S$ of $D(W^*)$ such that $W_k \in S$ infinitely often, then we have $W_k \to W^*$ with probability one as $k \to \infty$.* ∎

We denote $\lambda_1 > \lambda_2 > ... > \lambda_p \geq ... \geq \lambda_n > 0$ as the generalized eigenvalues of $A$ with respect to $B$, and $\phi_i$ as the generalized eigenvector corresponding to $\lambda_i$ such that $\phi_1,...,\phi_n$ are orthonormal with respect to $B$. Let $\Phi=[\phi_1...\phi_n]$ and $\Lambda=\mathrm{diag}(\lambda_1,...,\lambda_n)$ denote the matrix of generalized eigenvectors and eigenvalues of $A$ with respect to $B$. Note that if $\phi_i$ is a generalized eigenvector, then $d_i\phi_i$ ($|d_i|=1$) is also a generalized eigenvector.

In the next two lemmas, we first prove that all the possible equilibrium points of the ODE (11) are up to an arbitrary permutation of the $p$ generalized eigenvectors of $A$ with respect to $B$ corresponding to the $p$ largest generalized eigenvalues. We next prove that all these equilibrium points of the ODE (11) are unstable equilibrium points, except for $[d_1\phi_1 ... d_n\phi_n]$, where $|d_i|=1$ for $i=1,...,p$.

**Lemma 2.** *For the ordinary differential equation (11), let A1 and A3 hold. Then $W=\Phi DP$ are equilibrium points of (11), where $D=[D_1|0]^T$ is a $nXp$ matrix with $D_1$ being a $pXp$ diagonal matrix with diagonal elements $d_i$ such that $|d_i|=1$ or $d_i=0$, and $P$ is a $nXn$ arbitrary permutation matrix.* ∎

**Lemma 3.** *Let A1 and A3 hold. Then $W=\Phi D$ (where $D=[D_1|0]^T$, $D_1=\mathrm{diag}(d_1,...,d_p)$, $|d_i|=1$) are stable equilibrium points of the ODE (11). In addition, $W=\Phi DP$ ($d_i=0$ for $i\leq p$ or $P\neq I$) are unstable equilibrium points of the ODE (11).* ∎

**Lemma 4.** *For the ordinary differential equation (11), let A1 and A3 hold. Then the points $W=\Phi D$ (where $D=[D_1|0]^T$, $D_1=\mathrm{diag}(d_1,...,d_p)$, $|d_i|=1$ for $i=1,...,p$) are asymptotically stable.* ∎

**Lemma 5.** *Let A1-A3 hold. Then there exists a uniform upper bound for $\eta_k$ such that $W_k$ is uniformly bounded w.p.1.* ∎

The convergence of alg. (10) can now be established by referring to Theorem 1 of Ljung. **Theorem 1.** *Let A1-A3 hold. Assume that with probability one the process $\{W_k\}$ visits infinitely often a compact subset of the domain of attraction of one of the asymptotically stable points $\Phi D$. Then with probability one*

$$\lim_{k\to\infty} W_k = \Phi D.$$

**Proof.** By Lemma 2, $\Phi D$ ($|d_i|=1$) are asymptotically stable points of the ODE (11). Since we assume that $\{W_k\}$ visits a compact subset of the domain of attraction of $\Phi D$ infinitely often, Lemma 1 then implies the theorem. ∎

## 5. EXPERIMENTAL RESULTS

We describe the performance of algorithms (8) and (10) with an example of online interference cancellation in a high-dimensional signal, in a digital mobile communication problem. The problem occurs when the desired user transmits a signal from a far distance to the receiver, while another user simultaneously transmits very near to the base. For common receivers, the quality of the received signal from the desired user is dominated by interference from the user close to the base. Due to the high rate and large dimension of the data, the system demands an accurate detection method for just a few data samples.

If we use conventional (numerical analysis) methods, signal detection will require a significant part of the time slot allotted to a receiver, accordingly reducing the effective communication rate. Adaptive generalized eigen-decomposition algorithms, on the other hand, allow the tracking of slow changes, and directly performs signal detection.

The details of the data model can be found in Zoltowski *et al.* (1996). In this application, the duration for each transmitted code is 127 μs, within which we have 10μs of signal and 117μs of interference. We take 10 frequency samples equi-spaced between –0.4MHz to +0.4MHz. Using 6 antennas, the signal and interference correlation matrices are of dimension 60X60 in the complex domain.

We use both algorithms (8) and (10) for the cancellation of the interference. Figure 1 shows the convergence of the principal generalized eigenvector and eigenvalue. The closed form solution is obtained after collecting all of the signal and interference samples. In order to measure the accuracy of the algorithms, we compute the direction cosine of the estimated principal generalized eigenvector and the generalized eigenvector computed by the conventional method. The optimum value is one. We also show the estimated principal generalized eigenvalue in Figure 1b. The results show that both algorithms converge after the $4^{th}$ bit of signal.

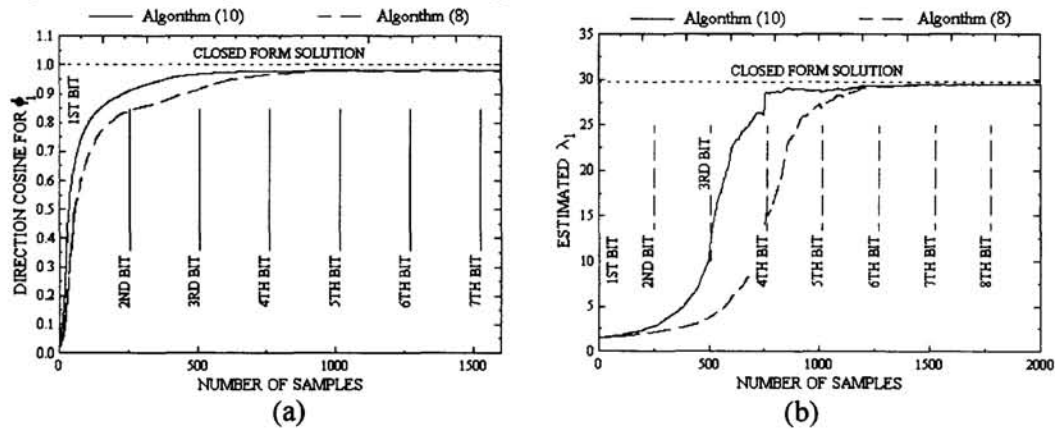

Figure 1. (a) Direction Cosine of Estimated First Principal Generalized Eigenvector, and (b) Estimated First Principal Generalized Eigenvalue.

## References

C.Chatterjee and V.Roychowdhury (1996), "Statistical Risk Analysis for Classification and Feature Extraction by Multilayer Perceptrons", *Proceedings IEEE Int'l Conference on Neural Networks*, Washington D.C.

K.Fukunaga (1990), *Introduction to Statistical Pattern Recognition*, 2nd Edition, New York: Academic Press.

P.Gallinari, S.Thiria, F.Badran, F.Fogelman-Soulie (1991), "On the Relations Between Discriminant Analysis and Multilayer Perceptrons", *Neural Networks*, Vol. 4, pp. 349-360.

L.Ljung (1977), "Analysis of Recursive Stochastic Algorithms", *IEEE Transactions on Automatic Control*, Vol. AC-22, No. 4, pp. 551-575.

A.R.Webb and D.Lowe (1990), "The Optimised Internal Representation of Multilayer Classifier Networks Performs Nonlinear Discriminant Analysis", *Neural Networks*, Vol. 3, pp. 367-375.

M.D.Zoltowski, C.Chatterjee, V.Roychowdhury and J.Ramos (1996), "Blind Adaptive 2D RAKE Receiver for CDMA Based on Space-Time MVDR Processing", submitted to *IEEE Transactions on Signal Processing*.
